# MAP Inference for
# Bayesian Inverse Reinforcement Learning

**Jaedeug Choi and Kee-Eung Kim**
bDepartment of Computer Science
Korea Advanced Institute of Science and Technology
Daejeon 305-701, Korea
jdchoi@ai.kaist.ac.kr, kekim@cs.kaist.ac.kr

## Abstract

The difficulty in inverse reinforcement learning (IRL) arises in choosing the best reward function since there are typically an infinite number of reward functions that yield the given behaviour data as optimal. Using a Bayesian framework, we address this challenge by using the maximum a posteriori (MAP) estimation for the reward function, and show that most of the previous IRL algorithms can be modeled into our framework. We also present a gradient method for the MAP estimation based on the (sub)differentiability of the posterior distribution. We show the effectiveness of our approach by comparing the performance of the proposed method to those of the previous algorithms.

## 1 Introduction

The objective of inverse reinforcement learning (IRL) is to determine the decision making agent's underlying reward function from its behaviour data and the model of environment [1]. The significance of IRL has emerged from problems in diverse research areas. In animal and human behaviour studies [2], the agent's behaviour could be understood by the reward function since the reward function reflects the agent's objectives and preferences. In robotics [3], IRL provides a framework for making robots learn to imitate the demonstrator's behaviour using the inferred reward function. In other areas related to reinforcement learning, such as neuroscience [4] and economics [5], IRL addresses the non-trivial problem of finding an appropriate reward function when building a computational model for decision making.

In IRL, we generally assume that the agent is an expert in the problem domain and hence it behaves optimally in the environment. Using the Markov decision process (MDP) formalism, the IRL problem is defined as finding the reward function that the expert is optimizing given the behaviour data of state-action histories and the environment model of state transition probabilities. In the last decade, a number of studies have addressed IRL in a direct (reward learning) and indirect (policy learning by inferring the reward function, *i.e.*, apprenticeship learning) fashions. Ng and Russell [6] proposed a sufficient and necessary condition on the reward functions that guarantees the optimality of the expert's policy and formulated a linear programming (LP) problem to find the reward function from the behaviour data. Extending their work, Abbeel and Ng [7] presented an algorithm for finding the expert's policy from its behaviour data with a performance guarantee on the learned policy. Ratliff *et al.* [8] applied the structured max-margin optimization to IRL and proposed a method for finding the reward function that maximizes the margin between the expert's policy and all other policies. Neu and Szepesvári [9] provided an algorithm for finding the policy that minimizes the deviation from the behaviour. Their algorithm unifies the direct method that minimizes a loss function of the deviation and the indirect method that finds an optimal policy from the learned reward function using IRL. Syed and Schapire [10] proposed a method to find a policy that improves the expert's policy using a game-theoretic framework. Ziebart *et al.* [11] adopted the principle of the

maximum entropy for learning the policy whose feature expectations are constrained to match those of the expert's behaviour. In addition, Neu and Szepesvári [12] provided a (non-Bayesian) unified view for comparing the similarities and differences among previous IRL algorithms.

IRL is an inherently ill-posed problem since there may be an infinite number of reward functions that yield the expert's policy as optimal. Previous approaches summarized above employ various preferences on the reward function to address the non-uniqueness. For example, Ng and Russell [6] search for the reward function that maximizes the difference in the values of the expert's policy and the second best policy. More recently, Ramachandran and Amir [13] presented a Bayesian approach formulating the reward preference as the prior and the behaviour compatibility as the likelihood, and proposed a Markov chain Monte Carlo (MCMC) algorithm to find the posterior mean of the reward function.

In this paper, we propose a Bayesian framework subsuming most of the non-Bayesian IRL algorithms in the literature. This is achieved by searching for the maximum-a-posteriori (MAP) reward function, in contrast to computing the posterior mean. We show that the posterior mean can be problematic for the reward inference since the loss function is integrated over the entire reward space, even including those inconsistent with the behaviour data. Hence, the inferred reward function can induce a policy much different from the expert's policy. The MAP estimate, however, is more robust in the sense that the objective function (the posterior probability in our case) is evaluated on a single reward function. In order to find the MAP reward function, we present a gradient method using the differentiability result of the posterior, and show the effectiveness of our approach through experiments.

## 2 Preliminaries

### 2.1 MDPs

A Markov decision process (MDP) is defined as a tuple $\langle S, A, T, R, \gamma, \alpha \rangle$: $S$ is the finite set of states; $A$ is the finite set of actions; $T$ is the state transition function where $T(s, a, s')$ denotes the probability $P(s'|s, a)$ of changing to state $s'$ from state $s$ by taking action $a$; $R$ is the reward function where $R(s, a)$ denotes the immediate reward of executing action $a$ in state $s$, whose absolute value is bounded by $R_{max}$; $\gamma \in [0, 1)$ is the discount factor; $\alpha$ is the initial state distribution where $\alpha(s)$ denotes the probability of starting in state $s$. Using matrix notations, the transition function is denoted as an $|S||A| \times |S|$ matrix $\boldsymbol{T}$, and the reward function is denoted as an $|S||A|$-dimensional vector $\boldsymbol{R}$.

A policy is defined as a mapping $\pi : S \rightarrow A$. The value of policy $\pi$ is the expected discounted return of executing the policy and defined as $V^\pi = \mathbb{E}\left[\sum_{t=0}^\infty \gamma^t R(s_t, a_t)|\alpha, \pi\right]$ where the initial state $s_0$ is determined according to initial state distribution $\alpha$ and action $a_t$ is chosen by policy $\pi$ in state $s_t$. The value function of policy $\pi$ for each state $s$ is computed by $V^\pi(s) = R(s, \pi(s)) + \gamma \sum_{s' \in S} T(s, \pi(s), s') V^\pi(s')$ such that the value of policy $\pi$ is calculated by $V^\pi = \sum_s \alpha(s) V^\pi(s)$. Similarly, the $Q$-function is defined as $Q^\pi(s, a) = R(s, a) + \gamma \sum_{s' \in S} T(s, a, s') V^\pi(s')$. We can rewrite the equations for the value function and the $Q$-function in matrix notations as

$$\boldsymbol{V}^\pi = \boldsymbol{R}^\pi + \gamma \boldsymbol{T}^\pi \boldsymbol{V}^\pi, \quad \boldsymbol{Q}_a^\pi = \boldsymbol{R}^a + \gamma \boldsymbol{T}^a \boldsymbol{V}^\pi \tag{1}$$

where $\boldsymbol{T}^\pi$ is an $|S| \times |S|$ matrix with the $(s, s')$ element being $T(s, \pi(s), s')$, $\boldsymbol{T}^a$ is an $|S| \times |S|$ matrix with the $(s, s')$ element being $T(s, a, s')$, $\boldsymbol{R}^\pi$ is an $|S|$-dimensional vector with the $s$-th element being $R(s, \pi(s))$, $\boldsymbol{R}^a$ is an $|S|$-dimensional vector with the $s$-th element being $R(s, a)$, and $\boldsymbol{Q}_a^\pi$ is an $|S|$-dimensional vector with the $s$-th element being $Q^\pi(s, a)$.

An optimal policy $\pi^*$ maximizes the value function for all the states, and thus should satisfy the Bellman optimality equation: $\pi$ is an optimal policy if and only if for all $s \in S$, $\pi(s) \in \operatorname{argmax}_{a \in A} Q^\pi(s, a)$. We denote $V^* = V^{\pi^*}$ and $Q^* = Q^{\pi^*}$.

When the state space is large, the reward function is often linearly parameterized: $R(s, a) = \sum_{i=1}^d w_i \phi_i(s, a)$ with the basis functions $\phi_i : S \times A \rightarrow \mathbb{R}$ and the weight vector $\boldsymbol{w} = [w_1, w_2, \cdots, w_d]$. Each basis function $\phi_i$ has a corresponding basis value $V_i^\pi$ of policy $\pi : V_i^\pi = \mathbb{E}\left[\sum_{t=0}^\infty \gamma^t \phi_i(s_t, a_t)|\alpha, \pi\right]$.

We also assume that the expert's behaviour is given as the set $\mathcal{X}$ of $M$ trajectories executed by the expert's policy $\pi_E$, where the $m$-th trajectory is an $H$-step sequence of state-action pairs: $\{(s_1^m, a_1^m), (s_2^m, a_2^m), \cdots, (s_H^m, a_H^m)\}$. Given the set of trajectories, the value and the basis value of the expert's policy $\pi_E$ can be empirically estimated by

$$\hat{V}^E = \frac{1}{M} \sum_{m=1}^{M} \sum_{h=1}^{H} \gamma^{h-1} R(s_h^m, a_h^m), \quad \hat{V}_i^E = \frac{1}{M} \sum_{m=1}^{M} \sum_{h=1}^{H} \gamma^{h-1} \phi_i(s_h^m, a_h^m).$$

In addition, we can empirically estimate the expert's policy $\hat{\pi}_E$ and its state visitation frequency $\hat{\mu}_E$ from the trajectories:

$$\hat{\pi}_E(s, a) = \frac{\sum_{m=1}^{M} \sum_{h=1}^{H} \mathbf{1}_{(s_h^m = s \wedge a_h^m = a)}}{\sum_{m=1}^{M} \sum_{h=1}^{H} \mathbf{1}_{(s_h^m = s)}}, \quad \hat{\mu}_E(s) = \frac{1}{MH} \sum_{m=1}^{M} \sum_{h=1}^{H} \mathbf{1}_{(s_h^m = s)}.$$

In the rest of the paper, we use the notation $f(\boldsymbol{R})$ or $f(x; \boldsymbol{R})$ for function $f$ in order to be explicit that $f$ is computed using reward function $\boldsymbol{R}$. For example, the value function $V^\pi(s; \boldsymbol{R})$ denotes the value of policy $\pi$ for state $s$ using reward function $\boldsymbol{R}$.

## 2.2 Reward Optimality Condition

Ng and Russell [6] presented a necessary and sufficient condition for reward function $\boldsymbol{R}$ of an MDP to guarantee the optimality of policy $\pi$: $\boldsymbol{Q}_a^\pi(\boldsymbol{R}) \leq \boldsymbol{V}^\pi(\boldsymbol{R})$ for all $a \in A$. From the condition, we obtain the following corollary (although it is a succinct reformulation of the theorem in [6], the proof is provided in the supplementary material).

**Corollary 1** *Given an MDP\R $\langle S, A, T, \gamma, \alpha \rangle$, policy $\pi$ is optimal if and only if reward function $\boldsymbol{R}$ satisfies*

$$\left[ \boldsymbol{I} - (\boldsymbol{I}^A - \gamma \boldsymbol{T})(\boldsymbol{I} - \gamma \boldsymbol{T}^\pi)^{-1} \boldsymbol{E}^\pi \right] \boldsymbol{R} \leq \boldsymbol{0}, \tag{2}$$

*where $\boldsymbol{E}^\pi$ is an $|S| \times |S||A|$ matrix with the $(s, (s', a'))$ element being 1 if $s = s'$ and $\pi(s') = a'$, and $\boldsymbol{I}^A$ is an $|S||A| \times |S|$ matrix constructed by stacking the $|S| \times |S|$ identity matrix $|A|$ times.*

We refer to Equation (2) as the *reward optimality condition* w.r.t. policy $\pi$. Since the linear inequalities define the region of the reward functions that yield policy $\pi$ as optimal, we refer to the region bounded by Equation (2) as the *reward optimality region* w.r.t. policy $\pi$. Note that there exist infinitely many reward functions in the reward optimality region even including constant reward functions (*e.g.* $\boldsymbol{R} = c\mathbf{1}$ where $c \in [-R_{max}, R_{max}]$). In other words, even when we are presented with the expert's policy, there are infinitely many reward functions to choose from, including the degenerate ones. To resolve this non-uniqueness in solutions, IRL algorithms in the literature employ various preferences on reward functions.

## 2.3 Bayesian framework for IRL (BIRL)

Ramachandran and Amir [13] proposed a Bayesian framework for IRL by encoding the reward function preference as the prior and the optimality confidence of the behaviour data as the likelihood. We refer to their work as BIRL.

Assuming the rewards are i.i.d., the prior in BIRL is computed by

$$P(\boldsymbol{R}) = \prod_{s \in S, a \in A} P(\boldsymbol{R}(s, a)). \tag{3}$$

Various distributions can be used as the prior. For example, the uniform prior can be used if we have no knowledge about the reward function other than its range, and a Gaussian or a Laplacian prior can be used if we prefer rewards to be close to some specific values.

The likelihood in BIRL is defined as an independent exponential distribution analogous to the softmax function:

$$P(\mathcal{X}|\boldsymbol{R}) = \prod_{m=1}^{M} \prod_{h=1}^{H} P(a_h^m | s_h^m, \boldsymbol{R}) = \prod_{m=1}^{M} \prod_{h=1}^{H} \frac{\exp(\beta Q^*(s_h^m, a_h^m; \boldsymbol{R}))}{\sum_{a \in A} \exp(\beta Q^*(s_h^m, a; \boldsymbol{R}))} \tag{4}$$

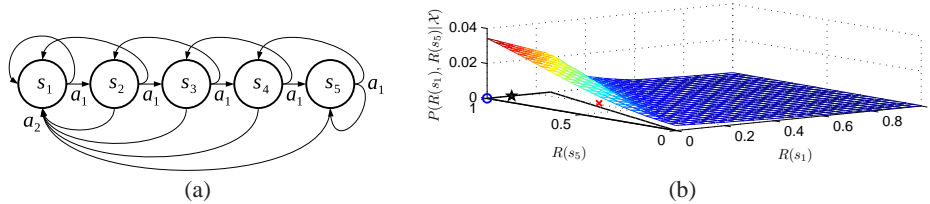

Figure 1: (a) 5-state chain MDP. (b) Posterior for $R(s_1)$ and $R(s_5)$ of the 5-state chain MDP.

where $\beta$ is a parameter that is equivalent to the inverse of temperature in the Boltzmann distribution.

The posterior over the reward function is then formulated by combining the prior and the likelihood, using Bayes theorem:

$$P(\boldsymbol{R}|\mathcal{X}) \propto P(\mathcal{X}|\boldsymbol{R})P(\boldsymbol{R}). \tag{5}$$

BIRL uses a Markov chain Monte Carlo (MCMC) algorithm to compute the posterior mean of the reward function.

## 3  MAP Inference in Bayesian IRL

In the Bayesian approach to IRL, the reward function can be determined using different estimates, such as the posterior mean, median, or maximum-a-posterior (MAP). The posterior mean is commonly used since it can be shown to be optimal under the mean square error function. However, the problem with the posterior mean in Bayesian IRL is that the error is integrated over the entire space of reward functions, even including infinitely many rewards that induce policies inconsistent with the behaviour data. This can yield a posterior mean reward function with an optimal policy again inconsistent with the data. On the other hand, the MAP does not involve an objective function that is integrated over the reward function space; it is simply a point that maximizes the posterior probability. Hence, it is more robust to infinitely many inconsistent reward functions. We present a simple example that compares the posterior mean and the MAP reward function estimation.

Consider an MDP with 5 states arranged in a chain, 2 actions, and the discount factor 0.9. As shown in Figure 1(a), we denote the leftmost state as $s_1$ and the rightmost state as $s_5$. Action $a_1$ moves to the state on the right with probability 0.6 and to the state on the left with probability 0.4. Action $a_2$ always moves to state $s_1$. The true reward of each state is $[0.1, 0, 0, 0, 1]$, hence the optimal policy chooses $a_1$ in every state. Suppose that we already know $R(s_2), R(s_3)$, and $R(s_4)$ which are all 0, and estimate $R(s_1)$ and $R(s_5)$ from the behaviour data $\mathcal{X}$ which contains optimal actions for all the states. We can compute the posterior $P(R(s_1), R(s_5)|\mathcal{X})$ using Equations (3), (4), and (5) under the assumption that $0 \leq \boldsymbol{R} \leq 1$ and priors $P(R(s_1))$ being $\mathcal{N}(0.1, 1)$, and $P(R(s_5))$ being $\mathcal{N}(1, 1)$. The reward optimality region can be also computed using Equation (2).

Figure 1(b) presents the posterior distribution of the reward function. The true reward, the MAP reward, and the posterior mean reward are marked with the black star, the blue circle, and the red cross, respectively. The black solid line is the boundary of the reward optimality region. Although the prior mean is set to the true reward, the posterior mean is outside the reward optimality region. An optimal policy for the posterior mean reward function chooses action $a_2$ rather than action $a_1$ in state $s_1$, while an optimal policy for the MAP reward function is identical to the true one. The situation gets worse when using the uniform prior. An optimal policy for the posterior mean reward function chooses action $a_2$ in states $s_1$ and $s_2$, while an optimal policy for the MAP reward function is again identical to the true one.

In the rest of this section, we additionally show that most of the IRL algorithms in the literature can be cast as searching for the MAP reward function in Bayesian IRL. The main insight comes from the fact that these algorithms try to optimize an objective function consisting of a regularization term for the preference on the reward function and an assessment term for the compatibility of the reward function with the behaviour data. The objective function is naturally formulated as the posterior in a Bayesian framework by encoding the regularization into the prior and the data compatibility into the likelihood. In order to subsume different approaches used in the literature, we generalize the

Table 1: IRL algorithms and their equivalent $f(\mathcal{X}; \boldsymbol{R})$ and prior for the Bayesian formulation. $q \in \{1, 2\}$ is for representing $L_1$ or $L_2$ slack penalties.

| Previous algorithm | $f(\mathcal{X}; \boldsymbol{R})$ | Prior |
|---|---|---|
| Ng and Russell's IRL from sampled trajectories [6] | $f_V$ | Uniform |
| MMP without the loss function [8] | $(f_V)^q$ | Gaussian |
| MWAL [10] | $f_G$ | Uniform |
| Policy matching [9] | $f_J$ | Uniform |
| MaxEnt [11] | $f_E$ | Uniform |

likelihood in Equation (4) to the following:

$$P(\mathcal{X}|\boldsymbol{R}) \propto \exp(\beta f(\mathcal{X}; \boldsymbol{R}))$$

where $\beta$ is a parameter for scaling the likelihood and $f(\mathcal{X}; \boldsymbol{R})$ is a function which will be defined appropriately to encode the data compatibility assessment used in each IRL algorithm. We then have the following result (the proof is provided in the supplementary material):

**Theorem 1** *IRL algorithms listed in Table 1 are equivalent to computing the MAP estimates with the prior and the likelihood using $f(\mathcal{X}; \boldsymbol{R})$ defined as follows:*

- $f_V(\mathcal{X}; \boldsymbol{R}) = \hat{V}^E(\boldsymbol{R}) - V^*(\boldsymbol{R})$
- $f_J(\mathcal{X}; \boldsymbol{R}) = -\sum_{s,a} \hat{\mu}_E(s) \left( J(s,a; \boldsymbol{R}) - \hat{\pi}_E(s,a) \right)^2$
- $f_G(\mathcal{X}; \boldsymbol{R}) = \min_i \left[ V_i^{\pi^*(\boldsymbol{R})} - \hat{V}_i^E \right]$
- $f_E(\mathcal{X}; \boldsymbol{R}) = \log \mathcal{P}_{MaxEnt}(\mathcal{X}|\boldsymbol{T}, \boldsymbol{R})$

*where $\pi^*(\boldsymbol{R})$ is an optimal policy induced by the reward function $\boldsymbol{R}$, $J(s,a; \boldsymbol{R})$ is a smooth mapping from reward function $\boldsymbol{R}$ to a greedy policy such as the soft-max function, and $\mathcal{P}_{MaxEnt}$ is the distribution on the behaviour data (trajectory or path) satisfying the principle of maximum entropy.*

The MAP estimation approach provides a rich framework for explaining the previous non-Bayesian IRL algorithms in a unified manner, as well as encoding various types of a priori knowledge into the prior distribution. Note that this framework can exploit the insights behind apprenticeship learning algorithms even if they do not explicitly learn a reward function (*e.g.*, MWAL [10]).

## 4 A Gradient Method for Finding the MAP Reward Function

We have proposed a unifying framework for Bayesian IRL and suggested that the MAP estimate can be a better solution to the IRL problem. We can then reformulate the IRL problem into the posterior optimization problem, which is finding $\boldsymbol{R}_{\mathrm{MAP}}$ that maximizes the (log unnormalized) posterior:

$$\boldsymbol{R}_{\mathrm{MAP}} = \mathrm{argmax}_{\boldsymbol{R}} P(\boldsymbol{R}|\mathcal{X}) = \mathrm{argmax}_{\boldsymbol{R}} \left[ \log P(\mathcal{X}|\boldsymbol{R}) + \log P(\boldsymbol{R}) \right]$$

Before presenting a gradient method for the optimization problem, we show that the generalized likelihood is differentiable almost everywhere.

The likelihood is defined for measuring the compatibility of the reward function $\boldsymbol{R}$ with the behaviour data $\mathcal{X}$. This is often accomplished using the optimal value function $\boldsymbol{V}^*$ or the optimal $Q$-function $\boldsymbol{Q}^*$ w.r.t. $\boldsymbol{R}$. For example, the empirical value of $\mathcal{X}$ is compared with $\boldsymbol{V}^*$ [6, 8], $\mathcal{X}$ is directly compared to the learned policy (*e.g.* the greedy policy from $\boldsymbol{Q}^*$) [9], or the probability of following the trajectories in $\mathcal{X}$ is computing using $\boldsymbol{Q}^*$ [13]. Thus, we generally assume that $P(\mathcal{X}|\boldsymbol{R}) = g(\mathcal{X}, \boldsymbol{V}^*(\boldsymbol{R}))$ or $g(\mathcal{X}, \boldsymbol{Q}^*(\boldsymbol{R}))$ where $g$ is differentiable w.r.t. $\boldsymbol{V}^*$ or $\boldsymbol{Q}^*$. The remaining question is the differentiability of $\boldsymbol{V}^*$ and $\boldsymbol{Q}^*$ w.r.t. $\boldsymbol{R}$, which we address in the following two theorems (The proofs are provided in the supplementary material.):

**Theorem 2** $\boldsymbol{V}^*(\boldsymbol{R})$ *and* $\boldsymbol{Q}^*(\boldsymbol{R})$ *are convex.*

**Theorem 3** $\boldsymbol{V}^*(\boldsymbol{R})$ *and* $\boldsymbol{Q}^*(\boldsymbol{R})$ *are differentiable almost everywhere.*

Theorems 2 and 3 relate to the previous work on gradient methods for IRL. Neu and Szepesvári [9] showed that $\boldsymbol{Q}^*(\boldsymbol{R})$ is Lipschitz continuous, and except on a set of measure zero (almost everywhere), it is Fréchet differentiable by Rademacher's theorem. We have obtained the same result

based on the reward optimality region, and additionally identified the condition for which $V^*(\boldsymbol{R})$ and $\boldsymbol{Q}^*(\boldsymbol{R})$ are non-differentiable (refer to the proof for details). Ratliff *et al.* [8] used a subgradient of their objective function because it involves differentiating $V^*(\boldsymbol{R})$. Using Theorem 3 for computing the subgradient of their objective function yields an identical result.

Assuming a differentiable prior, we can compute the gradient of the posterior using the result in Theorem 3 and the chain rule. If the posterior is convex, we will find the MAP reward function. Otherwise, as in [9], we will obtain a locally optimal solution. In the next section, we will experimentally show that the locally optimal solutions are nonetheless better than the posterior mean in practice. This is due to the property that they are generally found within the reward optimality region w.r.t. the policy consistent with the behaviour data.

The gradient method uses the update rule $\boldsymbol{R}_{\text{new}} \leftarrow \boldsymbol{R} + \delta_t \nabla_{\boldsymbol{R}} P(\boldsymbol{R}|\mathcal{X})$ where $\delta_t$ is an appropriate step-size (or learning rate). Since computing $\nabla_{\boldsymbol{R}} P(\boldsymbol{R}|\mathcal{X})$ involves computing an optimal policy for the current reward function and a matrix inversion, caching these results helps reduce repetitive computation. The idea is to compute the reward optimality region for checking whether we can reuse the cached result. If $\boldsymbol{R}_{\text{new}}$ is inside the reward optimality region of an already visited reward function $\boldsymbol{R}'$, they share the same optimal policy and hence the same $\nabla_{\boldsymbol{R}} V^\pi(\boldsymbol{R})$ or $\nabla_{\boldsymbol{R}} \boldsymbol{Q}^\pi(\boldsymbol{R})$. Given policy $\pi$, the reward optimality region is defined by $\boldsymbol{H}^\pi = \boldsymbol{I} - (\boldsymbol{I}^A - \gamma\boldsymbol{T})(\boldsymbol{I} - \gamma\boldsymbol{T}^\pi)^{-1}\boldsymbol{E}^\pi$, and we can reuse the cached result if $\boldsymbol{H}^\pi \cdot \boldsymbol{R}_{\text{new}} \leq \boldsymbol{0}$. The gradient method using this idea is presented in Algorithm 1.

---

**Algorithm 1** Gradient method for MAP inference in Bayesian IRL

---

**Input:** MDP\R, behaviour data $\mathcal{X}$, step-size sequence $\{\delta_t\}$, number of iterations $N$
 1: Initialize $\boldsymbol{R}$
 2: $\pi \leftarrow \text{solveMDP}(\boldsymbol{R})$
 3: $\boldsymbol{H}^\pi \leftarrow \text{computeRewardOptRgn}(\pi)$
 4: $\Pi \leftarrow \{\langle\pi, \boldsymbol{H}^\pi\rangle\}$
 5: **for** $t = 1$ to $N$ **do**
 6:     $\boldsymbol{R}_{\text{new}} \leftarrow \boldsymbol{R} + \delta_t \nabla_{\boldsymbol{R}} P(\boldsymbol{R}|\mathcal{X})$
 7:     **if** isNotInRewardOptRgn($\boldsymbol{R}_{\text{new}}, \boldsymbol{H}^\pi$) **then**
 8:         $\langle\pi, \boldsymbol{H}^\pi\rangle \leftarrow \text{findRewardOptRgn}(\boldsymbol{R}_{\text{new}}, \Pi)$
 9:         **if** isEmpty($\langle\pi, \boldsymbol{H}^\pi\rangle$) **then**
10:             $\pi \leftarrow \text{solveMDP}(\boldsymbol{R}_{\text{new}})$
11:             $\boldsymbol{H}^\pi \leftarrow \text{computeRewardOptRgn}(\pi)$
12:             $\Pi \leftarrow \Pi \cup \{\langle\pi, \boldsymbol{H}^\pi\rangle\}$
13:         **end if**
14:     **end if**
15:     $\boldsymbol{R} \leftarrow \boldsymbol{R}_{\text{new}}$
16: **end for**

---

## 5  Experimental Results

The first set of experiments was conducted in $N \times N$ gridworlds [7]. The agent can move west, east, north, or south, but with probability 0.3, it fails and moves in a random direction. The grids are partitioned into $M \times M$ non-overlapping regions, so there are $(\frac{N}{M})^2$ regions. The basis function is defined by a 0-1 indicator function for each region. The linearly parameterized reward function is determined by the weight vector $\boldsymbol{w}$ sampled i.i.d. from a zero mean Gaussian prior with variance 0.1 and $|w_i| \leq 1$ for all $i$. The discount factor is set to 0.99.

We compared the performance of our gradient method to those of other IRL algorithms in the literature: Maximum Margin Planning (MMP) [8], Maximum Entropy (MaxEnt) [11], Policy Matching with natural gradient (NatPM) and the plain gradient (PlainPM) [9], and Bayesian Inverse Reinforcement Learning (BIRL) [13]. We executed our gradient method for finding MAP using three different choices of the likelihood: B denotes the BIRL likelihood, and E and J denote the likelihood with $f_E$ and $f_J$, respectively. For the Bayesian IRL algorithms (BIRL and MAP), two types of the prior are prepared: U denotes the uniform prior and G denotes the true Gaussian prior. We evaluated the performance of the algorithms using the difference between $V^*$ (the value of the expert's policy) and $V^L$ (the value of the optimal policy induced by the learned weight $\boldsymbol{w}^L$ measured on the true weight $\boldsymbol{w}^*$) and the difference between $\boldsymbol{w}^*$ and $\boldsymbol{w}^L$ using $L_2$ norm.

Table 2: Results in the gridworld problems.

| $|S|$ | $\|\boldsymbol{w}^* - \boldsymbol{w}^L\|_2$ | | | | | | $V^* - V^L$ | | | | | |
| | $24 \times 24$ | | | $32 \times 32$ | | | $24 \times 24$ | | | $32 \times 32$ | | |
| $\dim(\boldsymbol{w})$ | 36 | 144 | 576 | 64 | 256 | 1024 | 36 | 144 | 576 | 64 | 256 | 1024 |
|---|---|---|---|---|---|---|---|---|---|---|---|---|
| NatPM | 3.04 | 6.84 | 16.83 | 3.50 | 8.88 | 21.25 | 2.49 | 8.97 | 8.74 | 1.08 | 12.84 | 10.97 |
| PlainPM | 3.77 | 6.63 | 16.60 | 5.21 | 9.05 | 17.36 | 0.15 | 0.67 | 0.51 | 0.41 | 1.28 | 1.91 |
| MaxEnt | 6.05 | 11.98 | 22.11 | 7.91 | 15.48 | 25.52 | 0.33 | 0.60 | 0.60 | 0.95 | 2.22 | 2.91 |
| MMP | 0.85 | 1.26 | 2.38 | 0.83 | 1.61 | 3.17 | 10.74 | 16.32 | 13.72 | 13.58 | 10.59 | 8.87 |
| BIRL-U | 3.27 | 5.67 | n.a. | 3.78 | 7.89 | n.a. | 1.38 | 0.80 | n.a. | 0.35 | 2.24 | n.a. |
| BIRL-G | 0.86 | 1.36 | n.a. | 0.98 | 1.71 | n.a. | 2.21 | 0.54 | n.a. | 0.50 | 0.90 | n.a. |
| MAP-B-U | 4.45 | 8.46 | 13.87 | 5.68 | 10.50 | 18.21 | **0.13** | 0.57 | 1.06 | 1.63 | 1.34 | 2.17 |
| MAP-B-G | 0.83 | 1.30 | 2.40 | 0.94 | 1.62 | 3.17 | 0.16 | 0.45 | 0.40 | 0.41 | **0.77** | **0.87** |
| MAP-E-G | 0.83 | 1.22 | 2.33 | 0.76 | 1.53 | 3.13 | 0.19 | 0.44 | 0.42 | 0.43 | 1.29 | 1.88 |
| MAP-J-G | **0.48** | **1.10** | **2.30** | **0.65** | **1.51** | **3.11** | 0.17 | **0.42** | **0.37** | **0.38** | 0.90 | 1.21 |

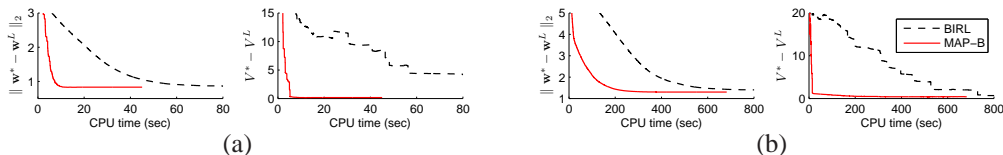

(a)             (b)

Figure 2: CPU timing results of BIRL and MAP-B in $24 \times 24$ gridworld problem. (a) $\dim(\boldsymbol{w}) = 36$. (b) $\dim(\boldsymbol{w}) = 144$.

We used training data with 10 trajectories of 50 time steps, collected from the simulated runs of the expert's policy. Table 2 shows the average performance over 10 training data. Most of the algorithms found the weight that induces an optimal policy whose performance is as good as that of the expert's policy (*i.e.*, small $V^* - V^L$) except for MMP and NatPM. The poor performance of MMP was due to the small size in the training data, as already noted in [14]. The poor performance of NatPM may be due to the ineffectiveness of pseudo-metric in high dimensional reward spaces, since PlainPM was able produce good performance. Regarding the learned weights, the algorithms using the true prior (MMP, BIRL, and the variants of MAP) found the weight close to the true one (*i.e.*, small $||\boldsymbol{w}^* - \boldsymbol{w}^L||_2$). Comparing BIRL and MAP-B is especially meaningful since they share the same prior and likelihood. The only difference was in computing the mean versus MAP from the posterior. MAP-B was consistently better than BIRL in terms of both $||\boldsymbol{w}^* - \boldsymbol{w}^L||_2$ and $V^* - V^L$. Finally, we note that the correct prior yields small $||\boldsymbol{w}^* - \boldsymbol{w}^L||_2$ and $V^* - V^L$ when we compare PlainPM, MaxEnt, BIRL-U, and MAP-B-U (uniform prior) to MAP-J-G, MAP-E-G, BIRL-G, and MAP-B-G (Gaussian prior), respectively.

Figure 2 compares the CPU timing results of the MCMC algorithm in BIRL and the gradient method in MAP-B for the 24×24 gridworld with 36 and 144 basis functions. BIRL takes much longer CPU time to converge than MAP-B since the former takes much larger number of iterations to converge, and in addition, each iteration requires solving an MDP with a sampled reward function. The CPU time gap gets larger as we increase the dimension of the reward function. Caching the optimal policies and gradients sped up the gradient method by factors of 1.5 to 4.2 until convergence, although not explicitly shown in the figure.

The second set of experiments was performed on a simplified car race problem, modified from [14]. The racetrack is shown in Figure 3. The shaded and white cells indicate the off-track and on-track locations, respectively. The state consists of the location and velocity of the car. The velocities in the vertical and horizontal directions are represented as 0, 1, or 2, and the net velocity is computed as the squared sum of directional velocities. The net velocity is regarded as high if greater than 2, zero if 0, and low otherwise. The car can increase, decrease, or maintain one of the directional velocities. The control of the car succeeds with $p=0.9$ if the net velocity is low, but $p=0.6$ if high. If the control fails, the velocity is maintained, and if the car attempts to move outside the racetrack, it remains in the previous location with velocity 0. The basis functions are 0-1 indicator functions for the goal locations, off-track locations, and 3 net velocity values (zero, low, high) while the car is on track. Hence, there are 3150 states, 5 actions, and 5 basis functions. The discount factor is set to 0.99.

Table 3: True and learned weights in the car race problem.

| | Goal | Off-track | Velocity while on track | | |
| --- | --- | --- | --- | --- | --- |
| | | | Zero | Low | High |
| Fast expert | 1.00 | 0.00 | 0.00 | 0.00 | 0.10 |
| BIRL | 0.96$\pm$0.02 | -0.20$\pm$0.03 | -0.04$\pm$0.01 | -0.12$\pm$0.02 | 0.32$\pm$0.02 |
| MAP-B | 1.00$\pm$0.00 | -0.19$\pm$0.02 | -0.03$\pm$0.01 | -0.13$\pm$0.01 | 0.29$\pm$0.01 |

Table 4: Statistics of the policies simulated in the car race problem.

| | Avg. steps | Avg. steps in locations | | Avg. steps in velocity | | |
| --- | --- | --- | --- | --- | --- | --- |
| | to goal | Off-track | On-track | Zero | Low | High |
| Fast expert | 20.41 | 1.56 | 17.85 | 2.01 | 3.40 | 12.44 |
| BIRL | 32.98$\pm$6.42 | 2.13$\pm$0.60 | 29.85$\pm$6.03 | 3.33$\pm$0.52 | 4.34$\pm$0.79 | 22.18$\pm$4.84 |
| MAP-B | 24.77$\pm$1.92 | 1.68$\pm$0.26 | 22.09$\pm$1.71 | 2.70$\pm$0.16 | 3.38$\pm$0.18 | 16.01$\pm$1.48 |

We designed two experts. The slow expert prefers low velocity and avoids the off-track locations, $\boldsymbol{w} = [1, -0.1, 0, 0.1, 0]$. The fast expert prefers high velocity, $\boldsymbol{w} = [1, 0, 0, 0, 0.1]$. We compared the posterior mean and the MAP using the prior $P(w_1)=\mathcal{N}(1, 1)$ and $P(w_2)=P(w_3)=P(w_4)=P(w_5)=\mathcal{N}(0, 1)$ assuming that we do not know the experts' preference on the locations nor the velocity, but we know the experts' ultimate goal is to reach one of the goal locations. We used BIRL for the posterior mean and MAP-B for the MAP estimation, hence using the identical prior and likelihood.

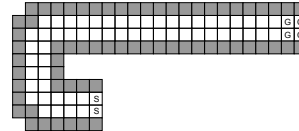

Figure 3: Racetrack.

We used 10 training data, each consisting of 5 trajectories. We omit the results regarding the slow expert since both BIRL and MAP-B successfully found the weight similar with the true one, which induced the slow expert's policy as optimal. However for the fast expert, MAP-B was significantly better than BIRL.[1] Table 3 shows the true and learned weights, and Table 4 shows some statistics characterizing the expert's and learned policies. The policy from BIRL tends to remain in high speed on the track for significantly more steps than the one from MAP-B since BIRL converged to a larger ratio of $w_5$ to $w_1$.

## 6 Conclusion

We have argued that, when using a Bayesian framework for learning reward functions in IRL, the MAP estimate is preferable over the posterior mean. Experimental results confirmed the effectiveness of our approach. We have also shown that the MAP estimation approach subsumes non-Bayesian IRL algorithms in the literature, and allows us to incorporate various types of a priori knowledge about the reward functions and the measurement of the compatibility with behaviour data.

We proved that the generalized posterior is differentiable almost everywhere, and proposed a gradient method to find a locally optimal solution to the MAP estimation. We provided the theoretical result equivalent to the previous work on gradient methods for non-Bayesian IRL, but used a different proof based on the reward optimality region.

Our work could be extended in a number of ways. For example, the IRL algorithm for partially observable environments in [15] mostly relies on Ng and Russell [6]'s heuristics for MDPs, but our work opens up new opportunities to leverage the insight behind other IRL algorithms for MDPs.

**Acknowledgments**

This work was supported by National Research Foundation of Korea (Grant# 2009-0069702) and the Defense Acquisition Program Administration and the Agency for Defense Development of Korea (Contract# UD080042AD)

## Footnotes

[1] All the results in Table 4 except for the average number of steps in the off-track locations are statistically significant at the 95% confidence level.

## References

[1] S. Russell. Learning agents for uncertain environments (extended abstract). In *Proceedings of COLT*, 1998.

[2] P. R. Montague and G. S. Berns. Neural economics and the biological substrates of valuation. *Neuron*, 36(2), 2002.

[3] B. D. Argall, S. Chernova, M. Veloso, and B. Browning. A survey of robot learning from demonstration. *Robotics and Autonomous Systems*, 57(5), 2009.

[4] Y. Niv. Reinforcement learning in the brain. *Journal of Mathematical Psychology*, 53(3), 2009.

[5] E. Hopkins. Adaptive learning models of consumer behavior. *Journal of Economic Behavior and Organization*, 64(3–4), 2007.

[6] A. Y. Ng and S. Russell. Algorithms for inverse reinforcement learning. In *Proceedings of ICML*, 2000.

[7] P. Abbeel and A. Y. Ng. Apprenticeship learning via inverse reinforcement learning. In *Proceedings of ICML*, 2004.

[8] N. D. Ratliff, J. A. Bagnell, and M. A. Zinkevich. Maximum margin planning. In *Proceedings of ICML*, 2006.

[9] G. Neu and C. Szepesvári. Apprenticeship learning using inverse reinforcement learning and gradient methods. In *Proceedings of UAI*, 2007.

[10] U. Syed and R. E. Schapire. A game-theoretic approach to apprenticeship learning. In *Proceedings of NIPS*, 2008.

[11] B. D. Ziebart, A. Maas, J. A. Bagnell, and A. K. Dey. Maximum entropy inverse reinforcement learning. In *Proceedings of AAAI*, 2008.

[12] G. Neu and C. Szepesvári. Training parsers by inverse reinforcement learning. *Machine Learning*, 77(2), 2009.

[13] D. Ramachandran and E. Amir. Bayesian inverse reinforcement learning. In *Proceedings of IJCAI*, 2007.

[14] A. Boularias and B. Chaib-Draa. Bootstrapping apprenticeship learning. In *Proceedings of NIPS*, 2010.

[15] J. Choi and K. Kim. Inverse reinforcement learning in partially observable environments. In *Proceedings of IJCAI*, 2009.

